# Parallel Optimization of Motion Controllers via Policy Iteration

**J. A. Coelho Jr., R. Sitaraman, and R. A. Grupen**
Department of Computer Science
University of Massachusetts, Amherst, 01003

## Abstract

This paper describes a policy iteration algorithm for optimizing the performance of a harmonic function-based controller with respect to a user-defined index. Value functions are represented as potential distributions over the problem domain, being control policies represented as gradient fields over the same domain. All intermediate policies are intrinsically safe, i.e. collisions are not promoted during the adaptation process. The algorithm has efficient implementation in parallel SIMD architectures. One potential application – travel distance minimization – illustrates its usefulness.

## 1 INTRODUCTION

Harmonic functions have been proposed as a uniform framework for the solution of several versions of the motion planning problem. Connolly and Grupen [Connolly and Grupen, 1993] have demonstrated how harmonic functions can be used to construct smooth, complete artificial potentials with no local minima. In addition, these potentials meet the criteria established in [Rimon and Koditschek, 1990] for *navigation functions*. This implies that the gradient of harmonic functions yields smooth ("realizable") motion controllers.

By construction, harmonic function-based motion controllers will always command the robot from any initial configuration to a goal configuration. The intermediate configurations adopted by the robot are determined by the boundary constraints and conductance properties set for the domain. Therefore, it is possible to tune both factors so as to extremize user-specified performance indices (e.g. travel time or energy) without affecting controller completeness.

Based on this idea, Singh et al. [Singh et al., 1994] devised a policy iteration method for combining two harmonic function-based control policies into a controller that minimized travel time on a given environment. The two initial control policies were

derived from solutions to two distinct boundary constraints (Neumann and Dirichlet constraints). The *policy space* spawned by the two control policies was parameterized by a mixing coefficient, that ultimately determined the obstacle avoidance behavior adopted by the robot. The resulting controller preserved obstacle avoidance, ensuring safety at every iteration of the learning procedure.

This paper addresses the question of how to adjust the conductance properties associated with the problem domain $\Omega$, such as to extremize an user-specified performance index. Initially, conductance properties are homogeneous across $\Omega$, and the resulting controller is optimal in the sense that it minimizes collision probabilities at every step [Connolly, 1994][1]. The method proposed is a policy iteration algorithm, in which the policy space is parameterized by the set of node conductances.

## 2   PROBLEM CHARACTERIZATION

The problem consists in constructing a path controller $\vec{\pi}_0$ that maximizes an integral performance index $\mathcal{P}$ defined over the set of all possible paths on a lattice for a closed domain $\Omega \subset \Re^n$, subjected to boundary constraints. The controller $\vec{\pi}_0$ is responsible for generating the sequence of configurations from an initial configuration $q_0$ on the lattice to the goal configuration $q_G$, therefore determining the performance index $\mathcal{P}$. In formal terms, the performance index $\mathcal{P}$ can be defined as follows:

**Def. 1**  *Performance index $\mathcal{P}$:*

$$\mathcal{P} = \sum_q \mathcal{P}_{q,\vec{\pi}} \quad for\ all\ q \in L(\Omega),\ where \quad \mathcal{P}_{q_0,\vec{\pi}} = \sum_{q=q_0}^{q_G} f(q).$$

$L(\Omega)$ *is a lattice over the domain* $\Omega$, $q_0$ *denotes an arbitrary configuration on* $L(\Omega)$, $q_G$ *is the goal configuration, and* $f(q)$ *is a function of the configuration* $q$.

For example, one can define $f(q)$ to be the available joint range associated with the configuration $q$ of a manipulator; in this case, $\mathcal{P}$ would be measuring the available joint range associated with all paths generated within a given domain.

### 2.1   DERIVATION OF REFERENCE CONTROLLER

The derivation of $\vec{\pi}_0$ is very laborious, requiring the exploration of the set of all possible paths. Out of this set, one is primarily interested in the subset of smooth paths. We propose to solve a simpler problem, in which the derived controller $\vec{\pi}$ is a numerical approximation to the optimal controller $\vec{\pi}_0$, and (1) generates smooth paths, (2) is admissible, and (3) locally maximizes $\mathcal{P}$. To guarantee (1) and (2), it is assumed that the control actions of $\vec{\pi}$ are proportional to the gradient of a harmonic function $\phi$, represented as the voltage distribution across a resistive lattice that tessellates the domain $\Omega$. The condition (3) is achieved through incremental changes in the set $G$ of internodal conductances; such changes maximize $\mathcal{P}$ locally.

**Necessary condition for optimality:** Note that $\mathcal{P}_{q_0,\vec{\pi}}$ defines a scalar field over $L(\Omega)$. It is assumed that there exists a well-defined neighborhood $\mathcal{N}(q)$ for node $q$; in fact, it is assumed that every node $q$ has two neighbors across each dimension. Therefore, it is possible to compute the gradient over the scalar field $\mathcal{P}_{q_0,\vec{\pi}}$ by locally approximating its rate of change across all dimensions. The gradient $\nabla \mathcal{P}_{q_0}$ defines

[1]This is exactly the control policy derived by the TD(0) reinforcement learning method, for the particular case of an agent travelling in a grid world with absorbing obstacle and goal states, and being rewarded only for getting to the goal states (see [Connolly, 1994]).

a *reference controller*, in the optimal situation, the actions of the controller $\vec{\pi}$ will parallel the actions of the reference controller. One can now formulate a policy iteration algorithm for the synthesis of the reference controller:

   *1. Compute $\vec{\pi} = -\vec{\nabla}\phi$, given conductances $G$;*
   *2. Evaluate $\vec{\nabla}\mathcal{P}_q$:*
      *– for each cell, compute $\mathcal{P}_{q,\vec{\phi}}$.*
      *– for each cell, compute $\vec{\nabla}\mathcal{P}_q$.*
   *3. Change $G$ incrementally, minimizing the approx. error $\epsilon = f(\vec{\pi}, \vec{\nabla}\mathcal{P}_q)$;*
   *4. If $\epsilon$ is below a threshold $\epsilon^0$, stop. Otherwise, return to (1).*

On convergence, the policy iteration algorithm will have derived a control policy that maximizes $\mathcal{P}$ globally, and is capable of generating smooth paths to the goal configuration. The key step on the algorithm is step (3), or how to reduce the current approximation error by changing the conductances $G$.

## 3   APPROXIMATION ALGORITHM

Given a set of internodal conductances, the approximation error $\epsilon$ is defined as

$$\epsilon \;=\; -\sum_{q \in L(\Omega)} \cos(\vec{\pi}, \vec{\nabla}\mathcal{P}) \tag{1}$$

or the sum over $L(\Omega)$ of the cosine of the angle between vectors $\vec{\pi}$ and $\vec{\nabla}\mathcal{P}$. The approximation error $\epsilon$ is therefore a function of the set $G$ of internodal conductances.

There exist $\Theta(nd^n)$ conductances in a $n$-dimensional grid, where $d$ is the discretization adopted for each dimension. Discrete search methods for the set of conductance values that minimizes $\epsilon$ are ruled out by the cardinality of the search space: $\Theta(k^{nd^n})$, if $k$ is the number of distinct values each conductance can assume. We will represent conductances as real values and use gradient descent to minimize $\epsilon$, according to the approximation algorithm below:

   *1. Evaluate the approximation error $\epsilon$;*
   *2. Compute the gradient $\vec{\nabla}\epsilon = \frac{\partial \epsilon}{\partial G}$;*
   *3. Update conductances, making $G = G - \alpha\vec{\nabla}\epsilon$;*
   *4. Normalize conductances, such that minimum conductance $g_{min} = 1$;*

Step (4) guarantees that every conductance $g \in G$ will be strictly positive. The conductances in a resistive grid can be normalized without constraining the voltage distribution across it, due to the linear nature of the underlying circuit. The complexity of the approximation algorithm is dominated by the computation of the gradient $\vec{\nabla}\epsilon(G)$. Each component of the vector $\vec{\nabla}\epsilon(G)$ can be expressed as

$$\frac{\partial \epsilon}{\partial g_i} = -\sum_{q \in L(\Omega)} \frac{\partial \cos(\vec{\pi}_q, \vec{\nabla}\mathcal{P}_q)}{\partial g_i}. \tag{2}$$

By assumption, $\vec{\pi}$ is itself the gradient of a harmonic function $\phi$ that describes the voltage distribution across a resistive lattice. Therefore, the calculation of $\frac{\partial \epsilon}{\partial g_i}$ involves the evaluation of $\frac{\partial \phi_q}{\partial g_i}$ over all domain $L(\Omega)$, or how the voltage $\phi_q$ is affected by changes in a certain conductance $g_i$.

For $n$-dimensional grids, $\frac{\partial \phi}{\partial g_i}$ is a matrix with $d^n$ rows and $\Theta(nd^n)$ columns. We posit that the computation of every element of $\frac{\partial \phi}{\partial g_i}$ is unnecessary: the effects of changing

$g_i$ will be more pronounced in a certain grid neighborhood of it, and essentially negligible for nodes beyond that neighborhood. Furthermore, this simplification allows for breaking up the original problem into smaller, independent sub-problems suitable to simultaneous solution in parallel architectures.

## 3.1   THE LOCALITY ASSUMPTION

The first simplifying assumption considered in this work establishes bounds on the neighborhood affected by changes on conductances at node $i$; specifically, we will assume that changes in elements of $g_i$ affect only the voltage at nodes in $\mathcal{N}(i)$, being $\mathcal{N}(i)$ the set composed of node $i$ and its direct neighbors. See [Coelho Jr. et al., 1995] for a discussion on the validity of this assumption. In particular, it is demonstrated that the effects of changing one conductance decay exponentially with grid distance, for infinite 2D grids. Local changes in resistive grids with higher dimensionality will be confined to even smaller neighborhoods.

The locality assumption simplifies the calculation of $\frac{\partial \epsilon}{\partial g_i}$ to

$$\frac{\partial \epsilon}{\partial g_i} = -\sum_{q \in \mathcal{N}(i)} \frac{\partial \cos(\vec{\pi}, \vec{\nabla P})}{\partial g_i} = -\sum_{q \in \mathcal{N}(i)} \frac{\partial}{\partial g_i} \left[ \frac{\vec{\pi} \cdot \vec{\nabla P}}{|\vec{\pi}||\vec{\nabla P}|} \right].$$

But

$$\frac{\partial}{\partial g_i} \left[ \frac{\vec{\pi} \cdot \vec{\nabla P}}{|\vec{\pi}||\vec{\nabla P}|} \right] = \frac{1}{|\vec{\pi}||\vec{\nabla P}|} \left[ \frac{\partial \vec{\pi}}{\partial g_i} \cdot \vec{\nabla P} - \frac{\vec{\pi} \cdot \vec{\nabla P}}{|\vec{\pi}|^2} \left( \frac{\partial \vec{\pi}}{\partial g_i} \cdot \vec{\pi} \right) \right].$$

Note that in the derivation above it is assumed that changes in $G$ affects primarily the control policy $\vec{\pi}$, leaving $\vec{\nabla P}$ relatively unaffected, at least in a first order approximation.

Given that $\vec{\pi} = -\vec{\nabla}\phi$, it follows that the component $\pi_j$ at node $q$ can be approximated by the change of potential across the dimension $j$, as measured by the potential on the corresponding neighboring nodes:

$$\pi_j|_q = \frac{\phi_{q-} - \phi_{q+}}{2\Delta^2}, \quad \text{and} \quad \frac{\partial \pi_j}{\partial g_i} = \frac{1}{2\Delta^2} \left[ \frac{\partial \phi_{q-}}{\partial g_i} - \frac{\partial \phi_{q+}}{\partial g_i} \right],$$

where $\Delta$ is the internodal distance on the lattice $L(\Omega)$.

## 3.2   DERIVATION OF $\frac{\partial \phi}{\partial g_i}$

The derivation of $\frac{\partial \phi}{\partial g_i}$ involves computing the Thévenin equivalent circuit for the resistive lattice, when every conductance $g$ connected to node $i$ is removed. For clarity, a 2D resistive grid was chosen to illustrate the procedure. Figure 1 depicts the equivalence warranted by Thévenin's theorem [Chua et al., 1987] and the relevant variables for the derivation of $\frac{\partial \phi_q}{\partial g_i}$. As shown, the equivalent circuit for the resistive grid consists of a four-port resistor, driven by four independent voltage sources. The relation between the voltage vector $\vec{\phi} = [\phi_1 \ \dots \ \phi_4]^T$ and the current vector $\vec{i} = [i_1 \ \dots \ i_4]^T$ is expressed as

$$\vec{\phi} = R\vec{i} + \vec{\omega}, \tag{3}$$

where $R$ is the impedance matrix for the grid equivalent circuit and $\vec{\omega}$ is the vector of open-circuit voltage sources. The grid equivalent circuit behaves exactly like the whole resistive grid; there is no approximation error.

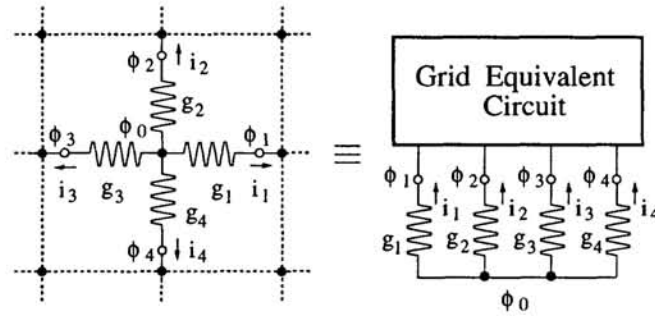

Figure 1: Equivalence established by Thévenin's theorem.

The derivation of the 20 parameters (the elements of $R$ and $\vec{\omega}$) of the equivalent circuit is detailed in [Coelho Jr. et al., 1995]; it involves a series of relaxation operations that can be efficiently implemented in SIMD architectures. The total number of relaxations for a grid with $n^2$ nodes is exactly $6n - 12$, or an average of $1/2n$ relaxations per link. In the context of this paper, it is assumed that $R$ and $\vec{\omega}$ are known. Our primary interest is to compute how changes in conductances $g_k$ affect the voltage vector $\vec{\phi}$, or the matrix

$$\frac{\partial \phi}{\partial g} = \left| \frac{\partial \phi_j}{\partial g_k} \right|, \quad \text{for} \quad \left\{ \begin{array}{rcl} j & = & 1, \dots, 4 \\ k & = & 1, \dots, 4. \end{array} \right.$$

The elements of $\frac{\partial \phi}{\partial g}$ can be computed by derivating each of the four equality relations in Equation 3 with respect to $g_k$, resulting in a system of 16 linear equations, and 16 variables – the elements of $\frac{\partial \phi}{\partial g}$. Notice that each element of $\vec{i}$ can be expressed as a linear function of the potentials $\vec{\phi}$, by applying Kirchhoff's laws [Chua et al., 1987]:

$$i_j = g_j \left[ \frac{\sum_{k=1}^{4} g_k \phi_k}{\sum_{k=1}^{4} g_k} - \phi_j \right].$$

## 4   APPLICATION EXAMPLE

A robot moves repeatedly toward a goal configuration. Its initial configuration is not known in advance, and every configuration is equally likely of being the initial configuration. The problem is to construct a motion controller that minimizes the overall travel distance for the whole configuration space. If the configuration space $\Omega$ is discretized into a number of cells, define the combined travel distance $D(\vec{\pi})$ as

$$D(\vec{\pi}) \quad = \quad \sum_{q \in L(\Omega)} d_{q,\vec{\pi}}, \qquad (4)$$

where $d_{q,\vec{\pi}}$ is the travel distance from cell $q$ to the goal configuration $q_G$, and robot displacements are determined by the controller $\vec{\pi}$. Figure 2 depicts an instance of the travel distance minimization problem, and the paths corresponding to its optimal solution, given the obstacle distribution and the goal configuration shown.

A resistive grid with $17 \times 17$ nodes was chosen to represent the control policies generated by our algorithm. Initially, the resistive grid is homogeneous, with all internodal resistances set to $1\Omega$. Figure 3 indicates the paths the robot takes when commanded by $\vec{\pi}^0$, the initial control policy derived from an homogeneous resistive grid.

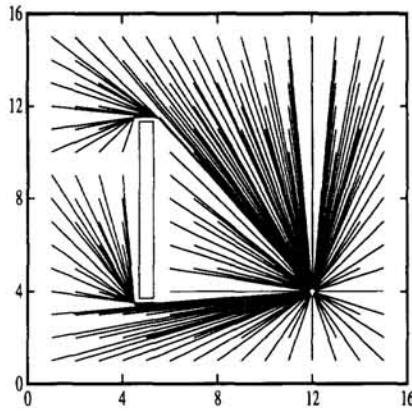

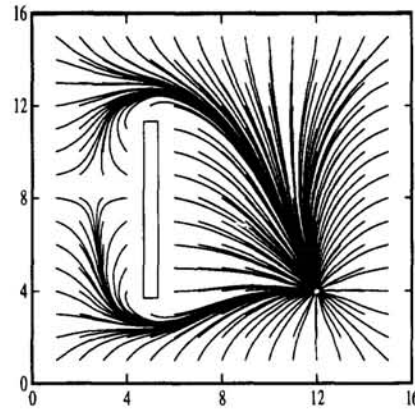

Figure 2: Paths for optimal solution of the travel distance minimization problem.

Figure 3: Paths for the initial solution of the same problem.

The conductances in the resistive grid were then adjusted over 400 steps of the policy iteration algorithm, and Figure 4 is a plot of the overall travel distance as a function of the number of steps. It also shows the optimal travel distance (horizontal line), corresponding to the optimal solution depicted in Figure 2. The plot shows that convergence is initially fast; in fact, the first 140 iterations are responsible for 90% of the overall improvement. After 400 iterations, the travel distance is within 2.8% of its optimal value. This residual error may be explained by the approximation incurred in using a discrete resistive grid to represent the potential distribution.

Figure 5 shows the paths taken by the robot after convergence. The final paths are straightened versions of the paths in Figure 3. Notice also that some of the final paths originating on the left of the I-shaped obstacle take the robot south of the obstacle, resembling the optimal paths depicted in Figure 2.

## 5   CONCLUSION

This paper presented a policy iteration algorithm for the synthesis of provably correct navigation functions that also extremize user-specified performance indices. The algorithm proposed solves the optimal feedback control problem, in which the final control policy optimizes the performance index over the whole domain, assuming that every state in the domain is as likely of being the initial state as any other state.

The algorithm modifies an existing harmonic function-based path controller by incrementally changing the conductances in a resistive grid. Departing from an homogeneous grid, the algorithm transforms an optimal controller (i.e. a controller that minimizes collision probabilities) into another optimal controller, that extremizes locally the performance index of interest. The tradeoff may require reducing the safety margin between the robot and obstacles, but collision avoidance is preserved at each step of the algorithm.

**Other Applications:** The algorithm presented can be used (1) in the synthesis of time-optimal velocity controllers, and (2) in the optimization of non-holonomic path controllers. The algorithm can also be a component technology for Intelligent Vehicle Highway Systems (IVHS), by combining (1) and (2).

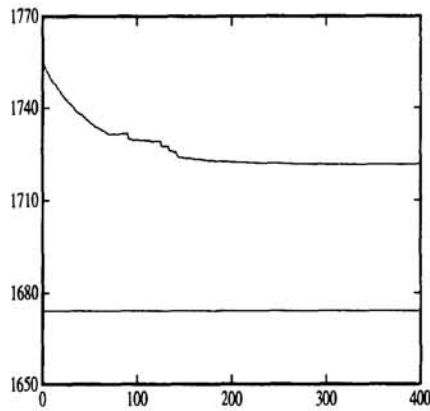
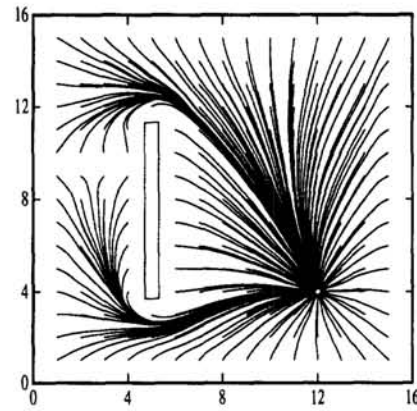

Figure 4: Overall travel distance, as a function of iteration steps.

Figure 5: Final paths, after 800 policy iteration steps.

**Performance on Parallel Architectures:** The proposed algorithm is computationally demanding; however, it is suitable for implementation on parallel architectures. Its sequential implementation on a SPARC 10 workstation requires $\approx$ 30 sec. per iteration, for the example presented. We estimate that a parallel implementation of the proposed example would require $\approx$ 4.3 ms per iteration, or 1.7 seconds for 400 iterations, given conservative speedups available on parallel architectures [Coelho Jr. et al., 1995].

### Acknowledgements

This work was supported in part by grants NSF CCR-9410077, IRI-9116297, IRI-9208920, and CNPq 202107/90.6.

## References

[Chua et al., 1987] Chua, L., Desoer, C., and Kuh, E. (1987). *Linear and Nonlinear Circuits*. McGraw-Hill, Inc., New York, NY.

[Coelho Jr. et al., 1995] Coelho Jr., J., Sitaraman, R., and Grupen, R. (1995). Control-oriented tuning of harmonic functions. Technical Report CMPSCI Technical Report 95-112, Dept. Computer Science, University of Massachusetts.

[Connolly, 1994] Connolly, C. I. (1994). Harmonic functions and collision probabilities. In *Proc. 1994 IEEE Int. Conf. Robotics Automat.*, pages 3015–3019. IEEE.

[Connolly and Grupen, 1993] Connolly, C. I. and Grupen, R. (1993). The applications of harmonic functions to robotics. *Journal of Robotic Systems*, 10(7):931–946.

[Rimon and Koditschek, 1990] Rimon, E. and Koditschek, D. (1990). Exact robot navigation in geometrically complicated but topologically simple spaces. In *Proc. 1990 IEEE Int. Conf. Robotics Automat.*, volume 3, pages 1937–1942, Cincinnati, OH.

[Singh et al., 1994] Singh, S., Barto, A., Grupen, R., and Connolly, C. (1994). Robust reinforcement learning in motion planning. In *Advances in Neural Information Processing Systems 6*, pages 655–662, San Francisco, CA. Morgan Kaufmann Publishers.